# Semi-supervised Learning with Weakly-Related Unlabeled Data: Towards Better Text Categorization

**Liu Yang**
Machine Learning Dept.
Carnegie Mellon University
5000 Forbes Avenue
Pittsburgh, PA 15213
liuy@cs.cmu.edu

**Rong Jin**
Dept. of Computer Sci. and Eng.
3115 Engineering Building
Michigan State University
East Lansing, MI 48824
rongjin@cse.msu.edu

**Rahul Sukthankar**
Intel Research Pittsburgh
and Carnegie Mellon Univ.
4720 Forbes Avenue, #410
Pittsburgh, PA 15213
rahuls@cs.cmu.edu

## Abstract

The cluster assumption is exploited by most semi-supervised learning (SSL) methods. However, if the unlabeled data is merely weakly related to the target classes, it becomes questionable whether driving the decision boundary to the low density regions of the unlabeled data will help the classification. In such case, the cluster assumption may not be valid; and consequently how to leverage this type of unlabeled data to enhance the classification accuracy becomes a challenge. We introduce "**Semi-supervised Learning with Weakly-Related Unlabeled Data**" (SSLW), an inductive method that builds upon the maximum-margin approach, towards a better usage of weakly-related unlabeled information. Although the SSLW could improve a wide range of classification tasks, in this paper, we focus on text categorization with a small training pool. The key assumption behind this work is that, even with different topics, the word usage patterns across different corpora tends to be consistent. To this end, SSLW estimates the optimal word-correlation matrix that is consistent with both the co-occurrence information derived from the weakly-related unlabeled documents and the labeled documents. For empirical evaluation, we present a direct comparison with a number of state-of-the-art methods for inductive semi-supervised learning and text categorization. We show that SSLW results in a significant improvement in categorization accuracy, equipped with a small training set and an unlabeled resource that is weakly related to the test domain.

## 1 Introduction

Semi-supervised Learning (**SSL**) takes advantage of a large amount of unlabeled data to enhance classification accuracy. Its application to text categorization is stimulated by the easy availability of an overwhelming number of unannotated web pages, in contrast to the limited number of annotated ones. Intuitively, corpora with different topics may not be content wise related, however, word usage exhibits consistent patterns within a language. Then the question is, what would be an effective SSL strategy to extract these valuable word usage patterns embedded in the unlabeled corpus? In this paper, we aim to identify a new data representation, that is on one hand informative to the target class (category), and on the other hand consistent with the feature coherence patterns exhibiting in the weakly related unlabeled data. We further turn it into a convex optimization problem, and solve it efficiently by an approximate approach. In this section, we first review the two types of semi-supervised learning: transductive SSL and inductive SSL. Then we state SSL with **weakly** related unlabeled data as a new challenge. Finally, we provide a strategy of how to address this challenge in the domain of text categorization, as well as a brief summary of related work in text categorization.

A variety of methods have been developed for transductive SSL [14, 21]. These methods can be grouped as: EM with generative mixture models, bootstrapping methods (Self-training, Co-training and the Yarowsky Algorithm), discriminative models (Transductive Support Vector Machines (TSVM) [2]) and data based methods, including Manifold Regularization [1], Information Regularization [17], and Low Density Separation(LDS) [11]. Specifically, TSVM extends the maximum margin principle of SVM to unlabeled data. It combines the regularization of SVMs on the labeled points with the cluster assumption on the unlabeled points, to enforce the decision boundary to lie in low density regions. Data based methods discover an inherent geometry in the data, and exploit it in finding a good classifier, to which additional regularization based on unlabeled data is added to avoid overfitting. Manifold Regularization uses the combinatorial Laplacian as a smoothness term. Based on the assumption that different classes usually form separate manifolds, it constructs decision functions that vary little along the data manifolds. Information Regularization seeks a good conditional $\Pr(y|x)$, assuming that the decision boundary lies in a low density area and $\Pr(y|x)$ only varies a little in the area of high density. Low Density Separation makes a similar assumption as Manifold Regularization and Information Regularization. In addition, it further computes a new data representation based on the unlabeled data, which often results in better classification performance for SSL.

Not many inductive SSL approaches have been presented. In general, the essential distinction between transductive learning and inductive learning is that transductive learning produces labels only for the available unlabeled data; while inductive learning not only produces labels for the unlabeled data, but also learns a classifier that can be used to predict labels for new data. In this sense, some SSL algorithms, though named as "transductive", have an inductive nature. For example, TSVM is an inductive learner, because it learns a classifier from a mixture of labeled and unlabeled data. Similarly, as an inductive component of Low Density Separation (LDS) [11], **$\Delta$ TSVMs** learns the SVM classification model in the primal, which can be used for predicting new data. However, the graph part of LDS is transductive, because the kernel and the graph distances are addressed by a prior eigen-decompostion and re-representation (MDS); thus, it is unclear how to make a prediction of a new test point other than by rebuilding the graph with the new test point. Manifold Regularization [1] also has an implementation with inductive nature. Harmonic Mixtures [22] is a recent work that aims to overcome the limitations of non-inductive inference. It models the data by a generative mixture of Gaussians, and adds discriminative regularization using the graph Laplacian.

In this paper, we focus on inductive SSL. In contrast to previous work in this area, we focus on the following important problem that has been overlooked before. As stated in [11], either directly or indirectly, all successful semi-supervised algorithms typically make the cluster assumption, which puts the decision boundary in low density areas without crossing the high density regions. Note that the cluster assumption is only meaningful when the labeled and unlabeled data are somehow closely related. When the unlabeled data comes from arbitrary data sources, their input patterns may not be closely related to that of labeled ones. As a result, the labeled and unlabeled data could be well separated, which makes it difficult, if not impossible, to exploit the cluster assumption. Hence, the key challenge is how to leverage the seemingly unrelated unlabeled data to improve the classification accuracy of the target classes. Analogous to transfer learning in which information from one category may be generalized to the others, we propose a scheme that helps the categorization of one data source, by making use of information from other unlabeled data sources with little relevance. Our study stands in contrast to the previous ones in that we aim to make maximum use of the unlabeled data that is weakly related to the test bed. We refer to this problem as "**SSL with weakly related unlabeled data**", or **SSLW** for short. We first build a maximum margin framework for SSL with **weakly** related unlabeled data. We then cast the framework into an Second Order Cone Programming (SOCP) problem that can be efficiently solved.

A typical approach for semi-supervised learning with weakly related unlabeled data, presented in the recent study [13] is to first derive a new data representation from unlabeled data, and then apply supervised learning technique to the derived new data representation. In [13], the authors proposed a SSL scheme termed as self-taught learning, which essentially conducts the unsupervised dimension reduction using sparse coding [10]. The new dimensions derived from the unlabeled data can then be used to represent the labeled data points for supervised learning. Notably, self-taught learning [13] performs coding and classification in two separate stages. In contrast, in our method, the construction of a good data representation is combined with the training of a maximum margin classifier under a unified framework. In particular, the data representation generated by our method

exploits both labeled and unlabeled data, which differentiates the proposed framework from self-taught learning.

In general, SSLW could improve a wide range of classification tasks. However in this study, we focus on text categorization with a small training set. Text categorization has been actively studied in the communities of Web data mining, information retrieval and statistical learning [9, 20]. A number of statistical learning techniques have been applied to text categorization [19], including the K Nearest Neighbor approaches, decision trees, Bayesian classifiers, inductive rule learning, neural networks, support vector machines (SVM), and logistic regression. Empirical studies [7] have shown that support vector machines (SVM) is the leading technique for text categorization. Given the limited amount of labeled documents, the key of semi-supervised text categorization is to exploit the unlabeled documents. The popular implementations of semi-supervised SVMs in [8, 15] are considered to be state-of-the-art in text categorization.

For text categorization with a small training pool, it is very likely that a large portion of words used by the testing documents are unseen in the training set, which could lead to a poor estimation of the similarity between documents. If we can identify the coherence information of words (e.g., word correlation) from both the labeled and unlabeled documents, we will be able to more accurately estimate the document similarity, particularly for documents sharing few or no common words, thus improving the overall classification accuracy. A straightforward approach is to utilize the word co-occurrence information for computing document similarity. However, this straightforward approach may not serve the best interests of word correlation, because not all of the co-occurrence patterns are useful. Some co-occurrence patterns (e.g., co-occurrence with common words) do not reflect the semantic relations among words, and some are not related to the target class. Consequently, it is critical to identify a subset of co-occurrence patterns that are most informative to the target classification problems. To address this problem, SSLW explicitly estimates the optimal word-correlation matrix for the target document categorization problem. The rest of paper is organized as follows. Section 2 introduces the basic notations and gives a brief review of the SVM dualism. In Section 3, we propose the framework of SSL with weakly-related unlabeled data, followed by an efficient algorithm for its computation in Section 4. Section 5 evaluates SSLW; and in section 6 we provide some insights into the experimental evidence and discuss future work.

## 2   Preliminaries

We introduce the notation used throughout this paper and briefly review the SVM dual formulation. Denote $\mathcal{L} = \{(\mathbf{x}_1, y_1), \ldots, (\mathbf{x}_l, y_l)\}$ as the collection of labeled documents, where $y_i$ is $+1$ when document $x_i$ belongs to a given document category and $-1$ when it does not (text categorization problem for multi-labeled documents can be treated as a set of independent binary classification problems). Let $\mathcal{U} = \{\mathbf{x}_{l+1} \ldots, \mathbf{x}_n\}$ be the unlabeled collection of documents. Let $V$ denote the size of the vocabulary. Importantly, as an SSL task with weakly-related unlabeled data, $\mathcal{U}$ comes from some external resources that are weakly related to the test domain. To facilitate our discussion, we denote the document-word matrix on $\mathcal{L}$ by $D = (\mathbf{d}_1, \mathbf{d}_2, \ldots, \mathbf{d}_l)$, where $\mathbf{d}_i \in \mathbb{N}^V$ represents the word-frequency vector for document $\mathbf{d}_i$. The **word-document matrix** on $\mathcal{L} + \mathcal{U}$ is denoted by $G = (\mathbf{g}_1, \mathbf{g}_2, \ldots, \mathbf{g}_V)$, where $\mathbf{g}_i = (g_{i,1}, g_{i,2}, \ldots, g_{i,n})$ represents the occurrence of the $i$th word in all the $n$ documents. Recall the dual formalism for SVM:

$$
\begin{aligned}
\max_{\boldsymbol{\alpha}} \quad & \boldsymbol{\alpha}^\top \mathbf{e} - \frac{1}{2}(\boldsymbol{\alpha} \circ \mathbf{y})^\top K(\boldsymbol{\alpha} \circ \mathbf{y}) \\
\text{s.t.} \quad & \boldsymbol{\alpha}^\top \mathbf{y} = 0 \\
& 0 \leq \alpha_i \leq C, \ i = 1, 2, \ldots, n,
\end{aligned} \tag{1}
$$

where $\boldsymbol{\alpha} = (\alpha_i, \alpha_2, \ldots, \alpha_n)$ are the weights assigned to the training documents, $\mathbf{e}$ is a vector with all elements being 1, and the symbol $\circ$ denotes an element-wise product between two vectors. $K \in \mathbb{R}^{n \times n}$ is the kernel matrix representing the document pairwise similarity and $K = D^\top D$.

## 3   The Framework of Semi-supervised Learning with Weakly-Related Unlabeled Data

In this section, we present the algorithm of Semi-supervised Learning with Weakly-Related Unlabeled Data (SSLW). As analysized in Section 1, the kernel similarity measure in the standard SVM

dual formalism $K = D^\top D$, is problematic in the sense that the similarity between two documents will be zero if they do not share any common words, even if there exists a pairwise relationship between the seen words and the unseen ones, from a large collection of documents. To solve this problem, we take into account a word-correlation matrix when computing the kernel similarity matrix, and we search for an optimal word-correlation matrix, towards maximizing the categorization margin. Specifically, we define the kernel matrix as $K = D^\top R D$, by introducing the word-correlation matrix $R \in \mathbb{R}^{V \times V}$, where each element $R_{i,j}$ represents the correlation between the $i$th and the $j$th words. Note $G^\top G$ is not a desirable solution to $R$, because it is improper to assign a high correlation to two words simply because of their high co-occurrence; the two words may be not closely related as judged by the maximum-margin criterion. Therefore, it is important to search for the optimal word-correlation matrix $R$ in addition to the maximum discovered in Eqn. (1), to maximize the categorization margin. We denote the optimal value of the objective function in Eqn. (1) as $\kappa(K)$:

$$\kappa(K) = \max_{\boldsymbol{\alpha}} \; \boldsymbol{\alpha}^\top \mathbf{e} - \frac{1}{2}(\boldsymbol{\alpha} \circ \mathbf{y})^\top K (\boldsymbol{\alpha} \circ \mathbf{y}) \tag{2}$$

Given the fact that $\kappa(K)$ is inversely-related to the categorization margin [4], minimizing $\kappa(K)$ is equivalent to maximizing the categorization margin.

Now we consider how to make maximum use of the weakly-related source $\mathcal{U}$. The $G$ matrix is crucial in capturing the word correlation information from the weakly-related external source $\mathcal{U}$. Thus, to incorporate the external source into the learning of the word-correlation matrix $R$, we regularize $R$ according to $G$ by introducing an internal representation of words $W = (\mathbf{w}_1, \mathbf{w}_2, \ldots, \mathbf{w}_V)$, where vector $\mathbf{w}_i$ is the internal representation of the $i$th word (This idea is similar to non-negative matrix factorization (NMF) [6]). We expect that $W$ carries an equivalent amount of information as $G$ does, i.e., $G$ and $W$ are roughly equivalent representations of words. As there exists a matrix $U$ such that the matrix $G$ can be recovered from $W$ by a linear transformation $G = UW$, the word-correlation matrix can be computed as $R = W^\top W$. Further, the constraints $G = UW$ and $R = WW^\top$ can be combined to obtain the following positive semi-definite constraint

$$\begin{pmatrix} R & G^\top \\ G & T \end{pmatrix} \succeq 0, \tag{3}$$

where $T = UU^\top$ [18]. Another strategy we use to involve the unlabeled data into the learning of word correlation, is to construct the word correlation matrix $R$ as a non-negative linear combination of the top $p$ right eigenvectors of $G$, i.e.,

$$R \;=\; \xi I_V + \sum_{i=1}^{p} (\alpha_i - \xi) \mathbf{s}_i \mathbf{s}_i^\top, \tag{4}$$

where $\{\mathbf{s}_i, i = 1, 2, \ldots, n\}$ denote the right eigenvectors of matrix $G$, sorted in descending order of their eigenvalues $\theta_i$. $I_V$ is the $V \times V$ identity matrix, and $\alpha_i \geq 0, i = 1, \ldots, p$ and $\xi \geq 0$ are non-negative combination weights. Note that introducing $\xi I_V$ ensures non-singularity of the matrix $R$, which is important when computing the expression for matrix $T$). This simplification of $R$ allows us to effectively extract and utilize the word co-occurrence information in the external source $\mathcal{U}$. Additionally, the positive semi-definite constraint $R \succeq 0$ is converted into simple non-negative constraints, i.e., $\xi \geq 0$ and $\{\alpha_i \geq 0\}_{i=1}^{p}$. The number of variables in $R$, which was originally $O(V^2)$, is now reduced to $p + 1$. A further insight into the combination weights reveals that, both the straightforward co-occurrence matrix $G^\top G$ and Manifold Regulization, give predefined weights for eigenvector combination and thus can be seen as the special cases of SSLW. Precisely speaking, the straightforward co-occurrence matrix $G^\top G$, directly uses the eigenvalues as the weights. Manifold Regularization does a slightly better job by defining the weights as a strict function of the eigenvalues. Different from both, we give SSLW the entire freedom to learn the weights from data. In this sense, SSLW generalizes these two methods.

Based on the above analysis, we reformulate an extension of SVM dual in Eqn. (1), to search for an optimal word-correlation matrix $R$, by exploiting the word co-occurrence information in the external $\mathcal{U}$, under maximum-margin criterion, i.e.,

$$\min_{R \in \Delta, U, W} \; \kappa(D^\top R D) \tag{5}$$

where the word-correlation matrix $R$ is restricted to domain $\Delta$ that is defined as

$$\Delta = \left\{ R \in \mathbf{S}_+^{V \times V} : \begin{pmatrix} R & G^\top \\ G & T \end{pmatrix} \succeq 0. \right\} \tag{6}$$

if we use (3) for $R$, and

$$\Delta = \left\{ R = \xi I_V + \sum_{i=1}^{p} (\alpha_i - \xi) \mathbf{s}_i \mathbf{s}_i^\top : \xi \geq 0, \alpha_i \geq 0, i = 1, \ldots, p \right\} \qquad (7)$$

if we use Eqn. (4) for $R$. Given the definition of $\kappa$ in Eqn. (2), Eqn. (5) is the following min-max problem without analytic solution.

$$\min_{R \in \Delta, U, W} \quad \max_{\boldsymbol{\alpha}} \; \boldsymbol{\alpha}^\top \mathbf{e} - \frac{1}{2} (\boldsymbol{\alpha} \circ \mathbf{y})^\top (D^\top R D)(\boldsymbol{\alpha} \circ \mathbf{y}) \qquad (8)$$

## 4  An Efficient Algorithm of SSLW

This section provides a computationally-efficient and scalable algorithm for solving the min-max problem in Eqn. (8), with domain $\Delta$ defined in (6). We first rewrite the maximization problem in Eqn. (1) into a minimization problem by computing its dual form:

$$\min_{t, \eta, \delta, \rho} \quad t + 2C\delta^\top \mathbf{e}$$

$$\text{s.t.} \quad \begin{pmatrix} K & \rho \circ \mathbf{y} + \lambda \mathbf{e} \\ (\rho \circ \mathbf{y} + \lambda \mathbf{e})^\top & t \end{pmatrix} \succeq 0$$

$$\rho = \mathbf{e} + \eta - \delta$$

$$\delta_i \geq 0, \; \eta_i \geq 0, \; i = 1, 2, \ldots, n. \qquad (9)$$

Then, by plugging Eqn. (9) back into Eqn. (5), we transform the min-max problem in Eqn. (8) into the following minimization problem:

$$\min_{t, \eta, \delta, \rho, R} \quad t + 2C\delta^\top \mathbf{e} + C_t \text{tr}(T) + C_r \text{tr}(R)$$

$$\text{s.t.} \quad \begin{pmatrix} D^\top R D & \rho \circ \mathbf{y} + \lambda \mathbf{e} \\ (\rho \circ \mathbf{y} + \lambda \mathbf{e})^\top & t \end{pmatrix} \succeq 0$$

$$\delta_i \geq 0, \; \eta_i \geq 0, \; i = 1, 2, \ldots, n$$

$$\rho = \mathbf{e} + \eta - \delta, \; \begin{pmatrix} R & G^\top \\ G & T \end{pmatrix} \succeq 0. \qquad (10)$$

Note that as our goal is to compute $R$ and $T$, thus any valid $(W, U)$ is sufficient, and no uniqueness constraints are imposed on $W$ and $U$.

In Eqn. (10), $C_t \text{tr}(T)$ and $C_r \text{tr}(R)$ serve as sparse regularizers for $R$ and $T$. They are added to improve the stability of the optimal solution, as well as to favor a simpler model over sophisticated ones. The parameters $C_t$ and $C_r$ are used to weigh the importance of the two regularization terms. The trace heuristic has been widely used to enforce a low-rank matrix by minimizing its trace in place of its rank. In the generalization of the trace heuristic presented by [5], the dual of the spectrum norm is the convex envelope of the rank on the set of matrices with norm less than one. The rank objective can be replaced with the dual of the spectral norm, for rank minimization. In other words, the best convex regularizer one can get for rank minimization is the trace function.

Eqn. (10) is a Semi-Definite Programming (SDP) problem [3], and in general can be solved using SDP packages such as SeDuMi [16]. However, solving a SDP problem is computationally expensive and does not easily scale to a large number of training examples. [18] recently provides an elegant scheme of rewriting a SDP problem into a Second Order Cone Programming (SOCP) problem that can be much more efficiently solved [3]. Technically, we adopt this procedure and rewrite Eqn. (10) into a typical SOCP problem that can be efficiently solved. Given the estimated word-correlation matrix $R$ and $K = D^\top R D$, the example weights $\alpha$ in SVM model can be estimated using the KKT conditions $\alpha = (\mathbf{y}\mathbf{y}^\top \circ K)^{-1}(\mathbf{e} + \eta - \delta + \lambda \mathbf{y})$. And the threshold $b$ in SVM can be obtained by solving the primal SVM using the linear programming technique.

## 5  Evaluation

In this section, we evaluate SSLW on text categorization with limited training data. The experiment set up is purely inductive, i.e., the testing feature space is invisible in the training phrase. As an SSL

task with weakly-related unlabeled data, the provided unlabeled data have little relevance to the test domain. We show that SSLW can achieve noticeable gains over the state-of-the-art methods in both inductive SSL and text categorization, and we provide insight into why this happens. Following [18], our implementation of SSLW selects the top $200$ right eigenvectors of the document-word matrix $G$ matrix to construct the $R$ matrix. As defined in Section 3, the $G$ matrix covers both the training sets and the weakly-related external collection.

**Evaluation datasets** Two standard datasets for text categorization are used as the evaluation test bed: the Reuters-21578 dataset and the WebKB dataset. For computational simplicity, $1000$ documents are randomly selected from the TREC AP88 dataset and are used as an external information source for both datasets. The AP88 dataset includes a collection of news documents reported by Associated Press in 1988. The same pre-processing and indexing procedure are applied to these three datasets, by using the Lemur Toolkit [1]. For the Reuters-21578 dataset, among the $135$ TOPICS categories, the $10$ categories with the largest amount of documents are selected (see Table 1). This results in a collection of $9,400$ documents. For the WebKB dataset, which has seven categories: student, faculty, staff, department, course, project, and other, we discard the category of "other" due to its unclear definition (see Table 2). This results in $4,518$ data samples in the selected dataset. The Reuters-21578 dataset and the TREC AP88 dataset have very limited relevance in topic; and the WebKB dataset and the TREC AP88 dataset are even less content-wise related.

| Category | earn | acq | money-fx | crude | grain | trade | interest | wheat | ship | corn |
|---|---|---|---|---|---|---|---|---|---|---|
| # Samples | 3987 | 2448 | 801 | 634 | 628 | 552 | 513 | 306 | 305 | 254 |

Table 1: The ten categories of the Reuters-21578 dataset with the largest amount of documents.

| Category | course | department | faculty | project | staff | student |
|---|---|---|---|---|---|---|
| # Samples | 930 | 182 | 1124 | 504 | 137 | 1641 |

Table 2: The six categories of the WebKB dataset.

**Evaluation Methodology** We focus on binary classification. For each class, **4** positive samples and **4** negative samples are randomly selected to form the training set; and the rest of the data serve as the testing set. As a rare classification problem, the testing data is very unbalanced. Therefore, we adopt the area under the ROC curve (AUR) [12] as the quantitative measurement of the binary classification performance for text categorization. AUR is computed based on the output of real-value scores of the classifiers returned for testing documents. Each experiment is repeated ten times, and the AUR averaged over these trials is reported.

**Baseline Methods** We use six baseline methods to demonstrate the strength of SSLW from different perspectives. The first two baselines are the standard SVM and the traditional TSVM. The third baseline is $\nabla$ **TSVM** [2], the inductive component of LDS, which delivers the state-of-the-art performance of SSL. The fourth baseline Manifold Regularization [3] (**ManifoldR** for short) is included as a state-of-the-art SSL approach with an inductive nature, and more importantly, being able to incorporate word relationship into the regularization. For the fifth baseline, we compare the word-correlation matrix estimated by SSLW, with the trivial word-correlation matrix $G^\top G$; and we name this baseline as **COR**. Finally, self-taught learning [13] serves as our sixth baseline method, named as **Self-taught**. It uses the unlabeled data to find an low-dimension representation, and then conducts standard classification in this new space.

**Text Categorization with Limited Training Data** We describe the AUR results of both the Reuters-21578 dataset and the WebKB datset, by using different methods. For the Reuters-21578 dataset, Table 3 summarizes the AUR comparison between the six baseline methods and SSLW. Both mean and variance of AUR are shown in the table. We observe that SSLW consistently outperforms the six baselines in AUR across most of the ten categories. In general, a t-test shows our performance gain is statistically significant compared to all the baselines at a significance level of $0.05$. Detailed analysis is provided below. First, TSVM and $\nabla$TSVM overall perform even worse than the standard SVM. This observation reveals that if the unlabeled data are only weakly relevant to the target class, it could

harm the categorization accuracy by simply pushing the decision boundary towards the low density regions, and away from the high density areas of the unlabeled data. It also justifies our intuitive hypothesis that the cluster assumption is not valid in this case. Second, the dramatic advantage of SSLW over the COR method confirms our previous analysis – learning a good word-correlation matrix that is jointly determined by the co-occurrence matrix and the classification margin (as SSLW does), can achieve significant gains over simply using the trivial form $G^\top G$. Third, we observe that SSLW algorithm consistently improves over Manifold Regularization, except on "trace" category where ManifoldR has a little advantage. Most noticeably, on "wheat" category and "ship" category, the AUR is improved by more than $10\%$, as a result of SSLW. These results demonstrate that SSLW is effective in improving text categorization accuracy with a small amount of training data. We also notice that, $\Delta$TSVM outperforms TSVM on some categories, but is slightly worse than TSVM on some others. The unstable performance of $\Delta$TSVM can possibly be explained by its gradient descent nature. Finally, our method receives gains against self-taught learning [13] on most categories. This proves SSLW is more effective than self-taught learning in using unlabeled data to improve classification. The gains can be attributed to the fact that Self-taught does coding and classification in two separate stages, while SSLW achieves these two purposes simultaneously.

A more careful examination indicates that SSLW also reduces the *standard deviation* in classification accuracy. The standard deviations by SSLW are mostly less than $2.5\%$; while those by the baseline methods are mostly above $2.5\%$. Over all the ten categories except the "money-fix" category, SSLW always delivers the lowest or the second lowest standard deviation, among all the six methods. We hypothesize that the large standard deviation by the baseline models is mainly due to the small number of training documents. In this situation, many words should only appear in a few training documents. As a result, the association between these words and the class labels can not be reliably established. In extreme cases where these words do not appear in any of the training documents, no association can be established between these words and the class labels. Evidently, test documents related to these unseen words are likely to be classified incorrectly. By contrast, SSLW can resolve this problem by estimating the word correlation. For a missing word, its association with the class label can be reliably estimated through the correlation with other words that appear frequently in the training examples.

Table 4 shows the AUR results of the WebKB dataset, from which we observe the similar trends as described above in the Reuters-21578 dataset. It is shown that SSLW maintains its clear advantage over the six baseline methods, across all the six categories.

| Category | SVM | TSVM | $\nabla$TSVM | ManifoldR | COR | Self-taught | SSLW |
|---|---|---|---|---|---|---|---|
| earn | $82.3 \pm 2.1$ | $70.9 \pm 4.1$ | $70.1 \pm 5.2$ | $86.4 \pm 2.1$ | $62.6 \pm 5.8$ | $65.9 \pm 3.5$ | $\mathbf{89.3 \pm 1.6}$ |
| acq | $69.7 \pm 3.0$ | $63.1 \pm 3.3$ | $59.2 \pm 4.1$ | $70.1 \pm 3.0$ | $51.2 \pm 4.7$ | $68.2 \pm 2.6$ | $\mathbf{73.5 \pm 3.3}$ |
| money-fx | $71.3 \pm 2.6$ | $67.4 \pm 3.1$ | $70.0 \pm 2.0$ | $74.0 \pm 2.6$ | $76.5 \pm 4.6$ | $75.7 \pm 3.9$ | $\mathbf{82.1 \pm 4.4}$ |
| crude | $69.7 \pm 3.3$ | $68.6 \pm 3.2$ | $59.9 \pm 4.7$ | $71.5 \pm 3.3$ | $56.0 \pm 5.7$ | $67.6 \pm 3.1$ | $\mathbf{77.5 \pm 1.7}$ |
| grain | $70.7 \pm 3.5$ | $68.7 \pm 2.3$ | $66.4 \pm 3.5$ | $75.1 \pm 3.5$ | $62.1 \pm 5.4$ | $69.0 \pm 2.9$ | $\mathbf{82.7 \pm 2.0}$ |
| trade | $82.7 \pm 3.4$ | $65.1 \pm 5.0$ | $71.5 \pm 4.2$ | $85.1 \pm 3.4$ | $78.8 \pm 5.2$ | $78.5 \pm 4.4$ | $\mathbf{84.4 \pm 3.9}$ |
| interest | $79.3 \pm 1.5$ | $60.2 \pm 3.9$ | $70.4 \pm 3.1$ | $85.0 \pm 1.5$ | $69.4 \pm 4.7$ | $76.5 \pm 2.5$ | $\mathbf{89.4 \pm 1.8}$ |
| wheat | $77.6 \pm 3.8$ | $61.9 \pm 3.6$ | $64.7 \pm 4.6$ | $79.1 \pm 3.8$ | $54.4 \pm 5.7$ | $67.1 \pm 2.6$ | $\mathbf{89.4 \pm 1.6}$ |
| ship | $70.4 \pm 2.6$ | $64.5 \pm 2.9$ | $65.8 \pm 3.9$ | $72.3 \pm 2.6$ | $52.1 \pm 5.0$ | $68.0 \pm 2.1$ | $\mathbf{82.8 \pm 1.4}$ |
| corn | $80.8 \pm 2.9$ | $65.4 \pm 2.1$ | $66.5 \pm 5.3$ | $77.0 \pm 5.0$ | $54.5 \pm 5.6$ | $66.8 \pm 3.7$ | $\mathbf{86.4 \pm 2.3}$ |

Table 3: The AUR results ($\%$) on the Reuters-21578 dataset with $8$ training examples per category.

| Category | SVM | TSVM | $\nabla$TSVM | ManifoldR | COR | Self-taught | SSLW |
|---|---|---|---|---|---|---|---|
| course | $66.8 \pm 2.2$ | $61.5 \pm 2.0$ | $61.8 \pm 2.9$ | $68.4 \pm 2.8$ | $63.3 \pm 5.4$ | $66.0 \pm 3.9$ | $\mathbf{76.2 \pm 2.5}$ |
| dept. | $72.2 \pm 2.8$ | $58.8 \pm 5.2$ | $63.7 \pm 3.5$ | $73.4 \pm 5.9$ | $58.3 \pm 5.1$ | $70.8 \pm 3.6$ | $\mathbf{87.6 \pm 2.2}$ |
| faculty | $56.7 \pm 3.4$ | $56.4 \pm 2.6$ | $54.2 \pm 3.0$ | $56.9 \pm 2.8$ | $53.1 \pm 4.6$ | $61.7 \pm 3.3$ | $\mathbf{61.6 \pm 3.4}$ |
| project | $59.6 \pm 2.9$ | $57.0 \pm 2.3$ | $60.3 \pm 1.4$ | $61.8 \pm 3.1$ | $50.0 \pm 5.9$ | $58.7 \pm 3.0$ | $\mathbf{69.5 \pm 3.2}$ |
| staff | $58.1 \pm 1.6$ | $53.0 \pm 1.1$ | $51.6 \pm 1.3$ | $52.9 \pm 0.9$ | $46.4 \pm 1.6$ | $59.9 \pm 1.9$ | $\mathbf{58.3 \pm 1.5}$ |
| student | $59.2 \pm 2.7$ | $54.0 \pm 2.3$ | $55.3 \pm 2.7$ | $59.4 \pm 3.1$ | $56.0 \pm 4.1$ | $61.0 \pm 1.9$ | $\mathbf{67.7 \pm 2.6}$ |

Table 4: The AUR results ($\%$) on the WebKB dataset with $8$ training examples per category.

# 6 Conclusion

This paper explores a new challenge in semi-supervised learning, i.e., how to leverage the unlabeled information that is weakly related to the target classes, to improve classification performance. We propose the algorithm of Semi-supervised Learning with Weakly-Related Unlabeled Data (SSLW) to address this challenge. SSLW extends the theory of support vector machines to effectively identify those co-occurrence patterns that are most informative to the categorization margin and ignore those that are irrelevant to the categorization task. Applied to text categorization with limited number of training samples, SSLW automatically estimates the word correlation matrix by effectively exploiting the word co-occurrence embedded in the weakly-related unlabeled corpus. Empirical studies show that SSLW significantly improves both the accuracy and the reliability of text categorization, given a small training pool and the additional unlabeled data that are weakly related to the test bed. Although SSLW is presented in the context of text categorization, it potentially facilitates classification tasks in a variety of domains. In future work, we will evaluate the benefits of SSLW on larger data sets and in other domains. We will also investigate SSLW's dependencies on the number of eigenvectors used, and its behavior when varying the number of labeled training examples.

**Acknowledgments**

The work was supported by the National Science Foundation (IIS-0643494) and National Institute of Health (1R01GM079688-01). Any opinions, findings, and conclusions or recommendations expressed in this material are those of the authors and do not necessarily reflect the views of NSF and NIH.

## Footnotes

[1] http://www.lemurproject.org/

[2] http://www.kyb.tuebingen.mpg.de/bs/people/chapelle/lds/

[3] http://manifold.cs.uchicago.edu/manifold_regularization/software.html

## References

[1] M. Belkin, P. Niyogi, and V. Sindhwani. Manifold regularization: A geometric framework for learning from labeled and unlabeled examples. Technical report, Univ. of Chicago, Dept. of Comp. Sci., 2004.

[2] K. Bennett and A. Demiriz. Semi-supervised support vector machines. In *Proc. NIPS*, 1998.

[3] S. Boyd and L. Vandenberghe. *Convex optimization*. Cambridge University Press, 2004.

[4] C. Burges. A tutorial on support vector machines for pattern recognition. *Data Mining and Knowledge Discovery*, 2(2), 1998.

[5] M. Fazel, H. Hindi, and S. Boyd. A rank minimization heuristic with application to minimum order system approximation. In *Proc. American Control Conf.*, 2001.

[6] P. O. Hoyer. Non-negative matrix factorization with sparseness constraints. *J. Mach. Learn. Res.*, 5, 2004.

[7] T. Joachims. Text categorization with support vector machines: learning with many relevant features. In *Proc. ECML*, 1998.

[8] T. Joachims. Transductive inference for text classification using support vector machines. In *Proc. ICML*, 1999.

[9] M. Lan, C. L. Tan, H.-B. Low, and S. Y. Sung. A comprehensive comparative study on term weighting schemes for text categorization with support vector machines. In *Proc. WWW*, 2005.

[10] H. Lee, A. Battle, R. Rajat, and A. Ng. Efficient sparse coding algorithms. In *Proc. NIPS*, 2007.

[11] A. Z. Olivier Chapelle. Semi-supervised classification by low density separation. In *Proc. Inter. Workshop on Artificial Intelligence and Statistics*, 2005.

[12] F. Provost, T. Fawcett, and R. Kohavi. The case against accuracy estimation for comparing induction algorithms. In *Proc. ICML*, 1998.

[13] R. Raina, A. Battle, H. Lee, B. Packer, and A. Y. Ng. Self-taught learning: transfer learning from unlabeled data. In *Proc. ICML*, 2007.

[14] M. Seeger. Learning with labeled and unlabeled data. Technical report, Univ. of Edinburgh, 2001.

[15] V. Sindhwani and S. S. Keerthi. Large scale semi-supervised linear support vector machines. In *Proc. ACM SIGIR*, 2006.

[16] J. F. Sturm. Using SeDuMi 1.02, a MATLAB toolbox for optimization over symmetric cones. *Optimization Methods Software*, 11/12(1–4), 1999.

[17] M. Szummer and T. Jaakkola. Information regularization with partially labeled data. In *Proc. NIPS*, 2002.

[18] L. Yang, R. Jin, C. Pantofaru, and R. Sukthankar. Discriminative cluster refinement: Improving object category recognition given limited training data. In *Proc. CVPR*, 2007.

[19] Y. Yang. An evaluation of statistical approaches to text categorization. *Journal of Info. Retrieval*, 1999.

[20] Y. Yang and J. O. Pedersen. A comparative study on feature selection in text categorization. In *Proc. ICML*, 1997.

[21] X. Zhu. Semi-supervised learning literature survey. Technical report, UW-Madison, Comp. Sci., 2006.

[22] X. Zhu, Z. Ghahramani, and J. Lafferty. Semi-supervised learning using gaussian fields and harmonic functions. In *Proc. ICML*, 2003.

